# Generalization by Weight-Elimination
# with Application to Forecasting

**Andreas S. Weigend**
Physics Department
Stanford University
Stanford, CA 94305

**David E. Rumelhart**
Psychology Department
Stanford University
Stanford, CA 94305

**Bernardo A. Huberman**
Dynamics of Computation
Xerox PARC
Palo Alto, CA 94304

## Abstract

Inspired by the information theoretic idea of minimum description length, we add a term to the back propagation cost function that penalizes network complexity. We give the details of the procedure, called weight-elimination, describe its dynamics, and clarify the meaning of the parameters involved. From a Bayesian perspective, the complexity term can be usefully interpreted as an assumption about prior distribution of the weights. We use this procedure to predict the sunspot time series and the notoriously noisy series of currency exchange rates.

## 1  INTRODUCTION

Learning procedures for connectionist networks are essentially statistical devices for performing inductive inference. There is a trade-off between two goals: on the one hand, we want such devices to be as general as possible so that they are able to learn a broad range of problems. This recommends large and flexible networks. On the other hand, the true measure of an inductive device is not how well it performs on the examples it has been shown, but how it performs on cases it has not yet seen, *i.e.*, its out-of-sample performance.

Too many weights of high precision make it easy for a net to fit the idiosyncrasies or "noise" of the training data and thus fail to generalize well to new cases. This *overfitting problem* is familiar in inductive inference, such as polynomial curve fitting. There are a number of potential solutions to this problem. We focus here on the so-called minimal network strategy. The underlying hypothesis is: if several nets fit the data equally well, the simplest one will on average provide the best generalization. Evaluating this hypothesis requires *(i)* some way of measuring simplicity and *(ii)* a search procedure for finding the desired net.

The complexity of an algorithm can be measured by the length of its minimal description

in some language. Rissanen [Ris89] and Cheeseman [Che90] formalized the old but vague intuition of Occam's razor as the information theoretic *minimum description length (MDL) criterion:* Given some data, the most probable model is the model that minimizes

$$\underbrace{\text{description length}}_{\text{cost}} = \underbrace{\text{description length}(\text{data}|\text{model})}_{\text{error}} + \underbrace{\text{description length}(\text{model})}_{\text{complexity}} .$$

This sum represents the trade-off between residual error and model complexity. The goal is to find a net that has the lowest complexity while fitting the data adequately. The complexity is dominated by the number of bits needed to encode the weights. It is roughly proportional to the number of weights times the number of bits per weight. We focus here on the procedure of weight-elimination that tries to find a net with the smallest *number of weights*. We compare it with a second approach that tries to minimize the *number of bits per weight*, thereby creating a net that is not too dependent on the precise values of its weights.

## 2   WEIGHT-ELIMINATION

In 1987, Rumelhart proposed a method for finding minimal nets within the framework of back propagation learning. In this section we explain and interpret the procedure and, for the first time, give the details of its implementation. [1]

### 2.1   METHOD

The idea is indeed simple in conception: add to the usual cost function a term which counts the number of parameters, and *minimize the sum* of performance error and the number of weights by back propagation,

$$\sum_{k \in \mathcal{T}} \left(\text{target}_k - \text{output}_k\right)^2 + \lambda \sum_{i \in \mathcal{C}} \frac{w_i^2/w_0^2}{1 + w_i^2/w_0^2} \quad . \tag{1}$$

The first term measures the performance of the net. In the simplest case, it is the sum squared error over the set of training examples $\mathcal{T}$. The second term measures the size of the net. Its sum extends over all connections $\mathcal{C}$. $\lambda$ represents the relative importance of the complexity term with respect to the performance term.

The learning rule is then to change the weights according to the gradient of the *entire* cost function, continuously doing justice to the trade-off between error and complexity. This differs from methods that consider a set of fixed models, estimate the parameters for each of them, and then compare between the models by considering the number of parameters.

The complexity cost as function of $w_i/w_0$ is shown in Figure 1(b). The extreme regions of very large and very small weights are easily interpreted. For $|w_i| \gg w_0$, the cost of a weight approaches unity (times $\lambda$). This justifies the interpretation of the complexity term as a counter of significantly sized weights. For $|w_i| \ll w_0$, the cost is close to zero. "Large" and "small" are defined with respect to the scale $w_0$, a free parameter of the weight-elimination procedure that has to be chosen.

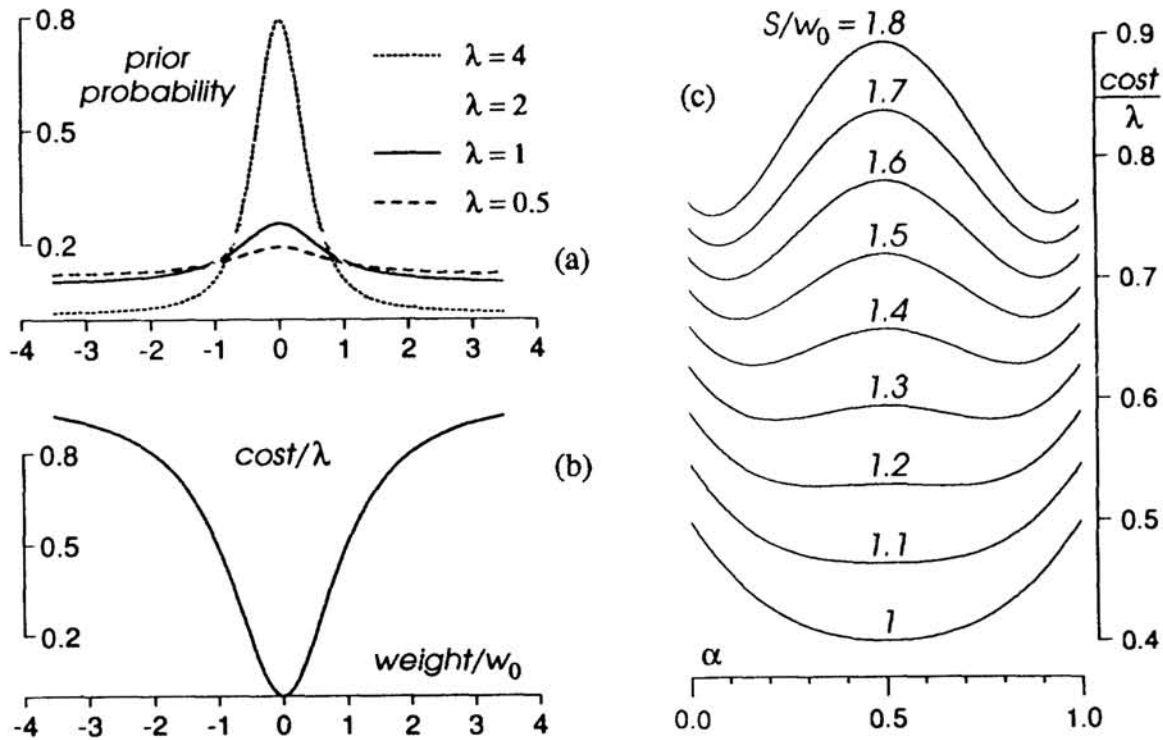

Figure 1: (a) Prior probability distribution for a weight. (b) Corresponding cost. (c) Cost for different values of $S/w_0$ as function of $\alpha = w_1/S$, where $S = w_1 + w_2$.

To clarify the meaning of $w_0$, let us consider a unit which is connected—redundantly—by two weights ($w_1$ and $w_2$) to the same signal source. Is it cheaper to have two smaller weights or just one large weight? Interestingly, as shown in Figure 1(c), the answer depends on the ratio $S/w_0$, where $S = w_1 + w_2$ is the relevant sum for the receiving unit. For values of $S/w_0$ up to about 1.1, there is only one minimum at $\alpha := w_1/S = 0.5$, *i.e.*, both weights are present and equal. When $S/w_0$ increases, this symmetry gets broken; it is cheaper to set one weight $\approx S$ and eliminate the other one.

Weight-*decay*, proposed by Hinton and by le Cun in 1987, is contained in our method of weight-*elimination* as the special case of large $w_0$. In the statistics community, this limit (cost $\propto w_i^2$) is known as *ridge regression*. The scale parameter $w_0$ thus allows us to express a preference for fewer large weights ($w_0$ small) or many small weights ($w_0$ large). In our experience, choosing $w_0$ of order unity is good for activations of order unity.

## 2.2   INTERPRETATION AS PRIOR PROBABILITY

Further insight can be gained by viewing the *cost as the negative log likelihood of the network, given the data*. In this framework[2], the error term is the negative logarithm of the probability of the data given the net, and the complexity term is the negative logarithm of the prior probability of the weights.

The cost function corresponds approximately to the assumption that the weights come from a mixture of two distributions. Relevant weights are drawn from a uniform distribution (to

allow for normalization of the probability, up to a certain maximum size). Weights that are merely the result of "noise" are drawn from a Gaussian-like distribution centered on zero; they are expected to be small. We show the prior probability for our complexity term for several values of $\lambda$ in Figure 1(a). If we wish to approximate the bump around zero by a Gaussian, its variance is given by $\sigma^2 = w_0^2/\lambda$. Its width scales with $w_0$.

Perhaps surprisingly the innocent weighting factor $\lambda$ now influences the width: the variance of the "noise" is inversely proportional to $\lambda$. The larger $\lambda$ is, the closer to zero a weight must be to have a reasonable probability of being a member of the "noise" distribution. Also, the larger $\lambda$ is, the more "pressure" small weights feel to become even smaller.

The following technical section describes how $\lambda$ is dynamically adjusted in training. From the perspective taken in Section 2.1, the usual increase of $\lambda$ during training corresponds to attaching more importance to the complexity term. From the perspective developed in this section, it corresponds to sharpening the peak of the weight distribution around zero.

## 2.3   DETAILS

Although the basic form of the weight-elimination procedure is simple, it is sensitive to the choice of $\lambda$.[3] If $\lambda$ is too small, it will have no effect. If $\lambda$ is too large, all of the weights will be driven to zero. Worse, a value of $\lambda$ which is useful for a problem that is easily learned may be too large for a hard problem, and a problem which is difficult in one region (at the start, for example) may require a larger value of $\lambda$ later on. We have developed some rules that make the performance relatively insensitive to the exact values of the parameters.

We start with $\lambda = 0$ so that the network can initially use all of its resources. $\lambda$ is changed after each epoch. It is usually gently incremented, sometimes decremented, and, in emergencies, cut down. The choice among these three actions depends on the value of the error on the training set $\overline{\mathcal{E}_n}$.

The subscript $n$ denotes the number of the epoch that has just finished. (Note that $\mathcal{E}_n$ is only the first term of the cost function (Equation 1). Since gradient descent minimizes the sum of both terms, $\mathcal{E}_n$ by itself can decrease or increase.) $\mathcal{E}_n$ is compared to three quantities, the first two derived from previous values of that error itself, the last one given externally:

- $\mathcal{E}_{n-1}$ Previous error.
- $A_n$   Average error (exponentially weighted over the past).
  It is defined as $A_n = \gamma A_{n-1} + (1 - \gamma)\mathcal{E}_n$ (with $\gamma$ relatively close to 1).
- $\mathcal{D}$   Desired error, the externally provided performance criterion.
  The strategy for choosing $\mathcal{D}$ depends on the specific problem. For example, "solutions" with an error larger than $\mathcal{D}$ might not be acceptable. Or, we may have observed (by monitoring the out-of-sample performance during training) that overfitting starts when a certain in-sample error is reached. Or, we may have some other estimate of the amount of noise in the training data. For toy problems, derived from approximating analytically defined functions (where perfect performance on the training data can be expected), a good choice is $\mathcal{D} = 0$. For hard problems, such as the prediction of currency exchange rates, $\mathcal{D}$ is set just below the error that corresponds to chance performance, since overfitting would occur if the error was reduced further.

After each epoch in training, we evaluate whether $\mathcal{E}_n$ is above or below each of these quantities. This gives eight possibilities. Three actions are possible:

- $\lambda \leftarrow \lambda + \Delta\lambda$
  In six cases, we increment $\lambda$ slightly. These are the situations in which things are going well: the error is already below than the criterion ($\mathcal{E}_n < \mathcal{D}$) and/or is still falling ($\mathcal{E}_n < \mathcal{E}_{n-1}$).

Incrementing $\lambda$ means attaching more importance to the complexity term and making the Gaussian a little sharper. Note that the primary parameter is actually $\Delta\lambda$. Its size is fairly small, of order $10^{-6}$.

In the remaining two cases, the error is worse than the criterion and it has grown compared to just before ($\mathcal{E}_n \geq \mathcal{E}_{n-1}$). The action depends on its relation to its long term average $\mathcal{A}_n$.

- $\lambda \leftarrow \lambda - \Delta\lambda$ $\qquad\qquad\qquad\qquad\qquad\qquad$ $\left[\text{ if } \mathcal{E}_n \geq \mathcal{E}_{n-1} \wedge \mathcal{E}_n < \mathcal{A}_n \wedge \mathcal{E}_n \geq \mathcal{D}\right]$

  In the less severe of those two cases, the performance is still improving with respect to the long term average ($\mathcal{E}_n < \mathcal{A}$). Since the error can have grown only slightly, we reduce $\lambda$ slightly.

- $\lambda \leftarrow 0.9\,\lambda$ $\qquad\qquad\qquad\qquad\qquad\qquad\quad$ $\left[\text{ if } \mathcal{E}_n \geq \mathcal{E}_{n-1} \wedge \mathcal{E}_n \geq \mathcal{A}_n \wedge \mathcal{E}_n \geq \mathcal{D}\right]$

  In this last case, the error has increased and exceeds its long term average. This can happen for two reasons. The error might have grown a lot in the last iteration. Or, it might not have improved by much in the whole period covered by the long term average, *i.e.*, the network might be trapped somewhere before reaching the performance criterion. The value of $\lambda$ is cut, hopefully prevent weight-elimination from devouring the whole net.

We have found that this set of heuristics for finding a minimal network while achieving a desired level of performance on the training data works rather well on a wide range of tasks. We give two examples of applications of weight-elimination. In the second example we show how $\lambda$ changes during training.

# 3   APPLICATION TO TIME SERIES PREDICTION

A central problem in science is predicting the future of temporal sequences; examples range from forecasting the weather to anticipating currency exchange rates. The desire to know the future is often the driving force behind the search for laws in science. The ability to forecast the behavior of a system hinges on two types of knowledge. The first and most powerful one is the knowledge of the laws underlying a given phenomenon. When expressed in the form of equations, the future outcome of an experiment can be predicted. The second, albeit less powerful, type of knowledge relies on the discovery of empirical regularities without resorting to knowledge of the underlying mechanism. In this case, the key problem is to determine which aspects of the data are merely idiosyncrasies and which aspects are truly indicators of the intrinsic behavior. This issue is particularly serious for real world data, which are limited in precision and sample size. We have applied nets with weight-elimination to time series of sunspots and currency exchange rates.

## 3.1   SUNSPOT SERIES [4]

When applied to predict the famous yearly sunspot averages, weight-elimination reduces the number of hidden units to three. Just having a small net, however, is not the ultimate goal: predictive power is what counts. The net has one half the out-of-sample error (on iterated single step predictions) of the benchmark model by Tong [Ton90].

What happens when we enlarge the input size from twelve, the optimal size for the benchmark model, to four times that size? As shown in [WRH90], the performance does not deteriorate (as might have been expected from a less dense distribution of data points in higher dimensional spaces). Instead, the net manages to ignore irrelevant information.

## 3.2  CURRENCY EXCHANGE RATES [5]

We use daily exchange rates (or *prices* with respect to the US Dollar) for five currencies (German Mark (DM), Japanese Yen, Swiss Franc, Pound Sterling and Canadian Dollar) to predict the *returns* at day $t$, defined as

$$r_t := \ln \frac{p_t}{p_{t-1}} = \ln \left( 1 + \frac{p_t - p_{t-1}}{p_{t-1}} \right) \approx \frac{p_t - p_{t-1}}{p_{t-1}} \quad . \tag{2}$$

For small changes, the return is the difference to the previous day normalized by the price $p_{t-1}$. Since different currencies and different days of the week may have different dynamics, we pick for one day (Monday) and one currency (DM). We define the task to be to learn *Monday DM dynamics:* given exchange rate information through a Monday, predict the DM - US\$ rate for the following day.

The net has 45 inputs for past daily DM returns, 5 inputs for the present Monday's returns of all available currencies, and 11 inputs for additional information (trends and volatilities), solely derived from the original exchange rates. The *k day trend* at day $t$ is the mean of the returns of the $k$ last days, $\frac{1}{k} \sum_{t-k+1}^{t} r_t$. Similarly, the *k day volatility* is defined to be the standard deviation of the returns of the $k$ last days.

The inputs are fully connected to the *5 sigmoidal hidden units* with range $(-1, 1)$. The hidden units are fully connected to *two output units*. The first one is to predict the next day return, $r_{t+1}$. This is a linear unit, trained with quadratic error. The second output unit focuses on the *sign* of the change. Its target value is one when the price goes up and zero otherwise. Since we want the unit to predict the probability that the return is positive, we choose a sigmoidal unit with range $(0, 1)$ and minimize cross entropy error.

The central question is whether the net is able to extract any signal from the training set that generalizes to the test sets. The performance is given as function of training time in epochs in Figure 2. [6]

The result is that the out-of-sample prediction is *significantly better than chance.* Weight-elimination reliably extracts a signal that accounts for between 2.5 and 4.0 per cent of the variance, corresponding to a correlation coefficient of $0.21 \pm 0.03$ for both test sets. In contrast, nets without precautions against overfitting show hopeless out-of-sample performance almost before the training has started. Also, none of the control experiments (randomized series and time-reversed series) reaches any significant predictability.

The dynamics of weight-elimination, discussed in Section 2.3, is also shown in Figure 2. $\lambda$ first grows very slowly. Then, around epoch 230, the error reaches the performance

$$\mathbf{arv}_S = \frac{\sum_{k \in S} \left( \text{target}_k - \text{prediction}_k \right)^2}{\sum_{k \in S} \left( \text{target}_k - \text{mean}_S \right)^2} = \frac{1}{\sigma_S^2} \frac{1}{N_S} \sum_{k \in S} \left( r_k - \widehat{r}_k \right)^2 \quad . \tag{3}$$

The averaging (division by $N_S$, the number of observations in set $S$) makes the measure independent of the size of the set. The normalization (division by $\sigma_S^2$, the estimated variance of the data in $S$), removes the dependence on the dynamic range of the data. Since the mean of the returns is close to zero, the random walk hypothesis corresponds to $\mathbf{arv} = 1.0$.

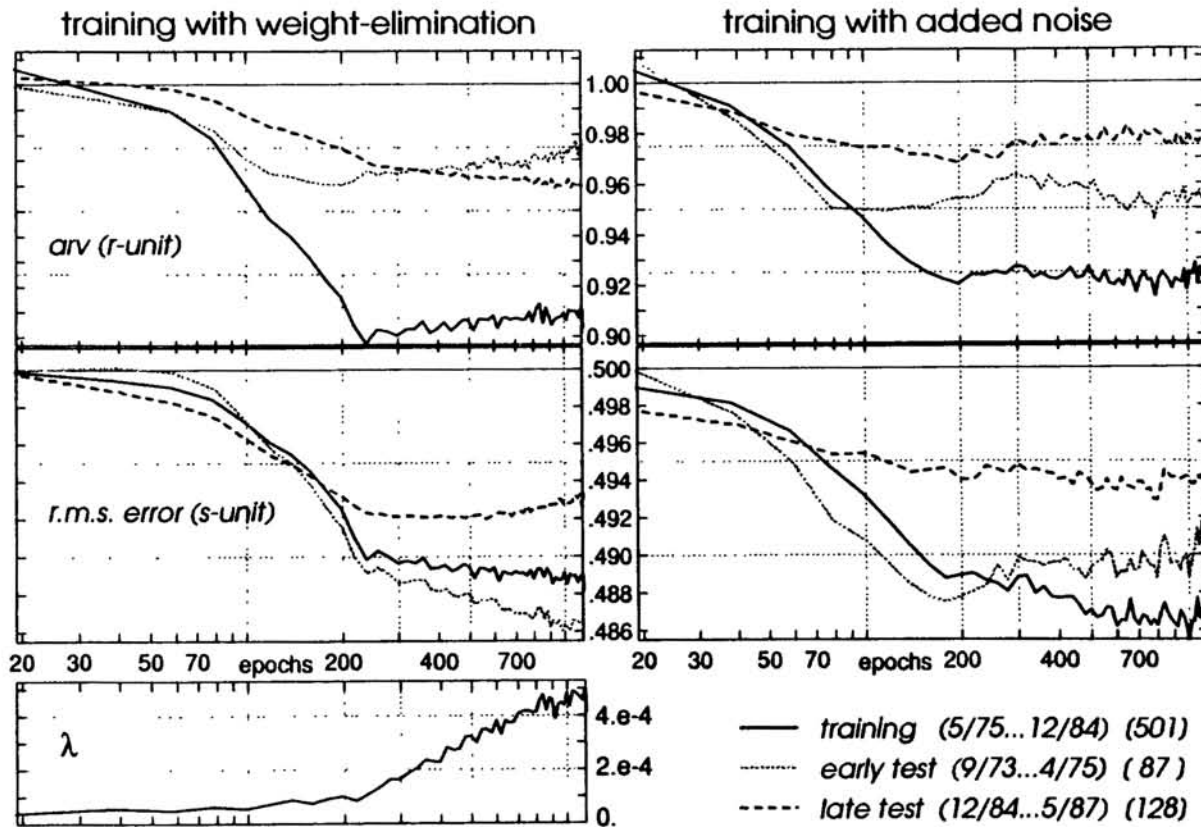

Figure 2: Learning curves of currency exchange rates for training with weight-elimination (left) and training with added noise (right). In-sample predictions are shown as solid lines, out-of sample predictions in grey and dashed. Top: average relative variance of the unit predicting the return (r-unit). Center: root-mean-square error of the unit predicting the sign (s-unit). Bottom: Weighting of the complexity term.

criterion. [7] The network starts to focus on the elimination of weights (indicated by growing $\lambda$) without further reducing its in-sample errors (solid lines), since that would probably correspond to overfitting.

We also compare training with weight-elimination with a method intended to make the parameters more robust. We add noise to the inputs, independently to each input unit, different at each presentation of each pattern.[8] This can be viewed as artificially enlarging the training set by smearing the data points around their centers. Smoother boundaries of the "basins of attraction" are the result. Viewed from the description length angle, it means saving bits by specifying the (input) weights with less precision, as opposed to eliminating some of them. The corresponding learning curves are shown on the right hand side of Figure 2. This simple method also successfully avoids overfitting.

Finally, we analyze the weight-eliminated network solution. The weights from the hidden units to the outputs are in a region where the complexity term acts as a counter. In fact only one or two hidden units remain. The weights from the inputs to the dead hidden units are also eliminated. For time series prediction, weight-elimination acts as hidden-unit elimination.

The weights between inputs and remaining hidden units are fairly small. Weight-elimination is in its quadratic region and prevents them from growing too large. Consequently, the activation of the hidden units lies in $(-0.4, 0.4)$. This prompted us to try a linear net where our procedure also works surprisingly well, yielding comparable performance to sigmoids.

Since all inputs are scaled to zero mean and unit standard deviation, we can gauge the importance of different inputs directly by the size of the weights. With weight-elimination, it becomes fairly clear which quantities are important, since connections that do not manage to reduce the error are not worth their price. A detailed description will be published in [WHR91]. Weight-elimination enhances the interpretability of the solution.

To summarize, we have a working procedure that finds small nets and can help prevent overfitting. With our rules for the dynamics of $\lambda$, weight-elimination is fairly stable. values of most parameters. In the examples we analyzed, the network manages to pick out some significant part of the dynamics underlying the time series.

## Footnotes

[1]The original formulation benefited from conversations with Paul Smolensky. Variations, and alternatives have been developed by Hinton, Hanson and Pratt, Mozer and Smolensky, le Cun, Denker and Solla, Ji, Snapp and Psaltis and others. They are discussed in Weigend [Wei91].

[2]This perspective is expanded in a forthcoming paper by Rumelhart *et al.* [RDGC92].

[3]The reason that $\lambda$ appears at all is because weight-elimination only deals with a part of the complete network complexity, and this only approximately. In a theory rigidly derived from the minimum description length principle, no such parameter would appear.

[4]We here only briefly summarize our results on sunspots. Details have been published in [WHR90] and [WRH90].

[5]We thank Blake LeBaron for sending us the data.

[6]The error of the unit predicting the return is expressed as the *average relative variance*

[7]Guided by cross-validation, we set the criterion (for the sum of the squared errors from both outputs) to 650. With this value, the choice of the other parameters is not critical, as long as they are fairly small. We used a learning rate of $2.5 \times 10^{-4}$, no momentum, and an increment $\Delta\lambda$ of $2.5 \times 10^{-6}$. If the criterion was set to zero, the balance between error and complexity would be fragile in such a hard problem.

[8]We add Gaussian noise with a rather large standard deviation of 1.5 times the signal. The exact value is not crucial: similar performance is obtained for noise levels between 0.7 and 2.0 .

# References

[Che90]    Peter C. Cheeseman. **On finding the most probable model.** In J. Shrager and P. Langley (eds.) *Computational Models of Scientific Discovery and Theory Formation*, p. 73. Morgan Kaufmann, 1990.

[RDGC92]  David E. Rumelhart, Richard Durbin, Richard Golden, and Yves Chauvin. **Backpropagation: theoretical foundations.** In Y. Chauvin and D. E. Rumelhart (eds.) *Backpropagation and Connectionist Theory*. Lawrence Erlbaum, 1992.

[Ris89]    Jorma Rissanen. **Stochastic Complexity in Statistical Inquiry.** World Scientific, 1989.

[Ton90]    Howell Tong. **Non-linear Time Series: a Dynamical System Approach.** Oxford University Press, 1990.

[Wei91]    Andreas S. Weigend. **Connectionist Architectures for Time Series Prediction.** PhD thesis, Stanford University, 1991. (in preparation)

[WHR90]    Andreas S. Weigend, Bernardo A. Huberman, and David E. Rumelhart. **Predicting the future: a connectionist approach.** *International Journal of Neural Systems*, 1:193, 1990.

[WHR91]    Andreas S. Weigend, Bernardo A. Huberman, and David E. Rumelhart. **Predicting sunspots and currency rates with connectionist networks.** In M. Casdagli and S. Eubank (eds.) *Proceedings of the 1990 NATO Workshop on Nonlinear Modeling and Forecasting (Santa Fe)*. Addison-Wesley, 1991.

[WRH90]    Andreas S. Weigend, David E. Rumelhart, and Bernardo A. Huberman. **Backpropagation, weight-elimination and time series prediction.** In D. S. Touretzky, J. L. Elman, T. J. Sejnowski, and G. E. Hinton (eds.) *Proceedings of the 1990 Connectionist Models Summer School*, p 105. Morgan Kaufmann, 1990.
